# Variational minimax estimation of discrete distributions under KL loss

**Liam Paninski**
Gatsby Computational Neuroscience Unit
University College London
liam@gatsby.ucl.ac.uk
http://www.gatsby.ucl.ac.uk/∼liam

## Abstract

We develop a family of upper and lower bounds on the worst-case expected KL loss for estimating a discrete distribution on a finite number $m$ of points, given $N$ i.i.d. samples. Our upper bounds are approximation-theoretic, similar to recent bounds for estimating discrete entropy; the lower bounds are Bayesian, based on averages of the KL loss under Dirichlet distributions. The upper bounds are convex in their parameters and thus can be minimized by descent methods to provide estimators with low worst-case error; the lower bounds are indexed by a one-dimensional parameter and are thus easily maximized. Asymptotic analysis of the bounds demonstrates the uniform KL-consistency of a wide class of estimators as $c = N/m \to \infty$ (no matter how slowly), and shows that no estimator is consistent for $c$ bounded (in contrast to entropy estimation). Moreover, the bounds are asymptotically tight as $c \to 0$ or $\infty$, and are shown numerically to be tight within a factor of two for all $c$. Finally, in the sparse-data limit $c \to 0$, we find that the Dirichlet-Bayes (add-constant) estimator with parameter scaling like $-c \log(c)$ optimizes both the upper and lower bounds, suggesting an optimal choice of the "add-constant" parameter in this regime.

## Introduction

The estimation of discrete distributions given finite data — "histogram smoothing" — is a canonical problem in statistics and is of fundamental importance in applications to language modeling, informatics, and safari organization (*1–3*). In particular, estimation of discrete distributions under Kullback-Leibler (KL) loss is of basic interest in the coding community, in the context of two-step universal codes (*4, 5*). The problem has received signicant attention from a variety of statistical viewpoints (see, e.g., (*6*) and references therein); in this work, we will focus on the "minimax" approach, that is, on developing estimators which work well even in the worst case, with the performance of an estimator measured by the average KL loss. The recent work of (*7*) and (*8*) has answered many of the important asymptotic questions in the heavily-sampled limit, where the number of data samples, $N$, is much larger than the number of support points, $m$, of the unknown distribution; in particular, the optimal (minimax) error rate has been identified in closed form in the case that $m$ is fixed and $N \to \infty$, and a simple estimator that asymptotically achieves this optimum

has been described. Our goal here is to analyze further the opposite case, when $N/m$ is bounded or even small (the sparse data case). It will turn out that the estimators which are asymptotically optimal as $N/m \to \infty$ are far from optimal in this sparse data case, which may be considered more important for applications to modeling of large dictionaries.

Much of our approach is influenced by the similarities to the entropy estimation problem (9–11), where the sparse data regime is also important for applications and of independent mathematical interest: how do we decide how much probability to assign to bins for which no samples, or very few samples, are observed? We will emphasize the similarities (and important differences) between these two problems throughout.

## Upper bounds

The basic idea is to find a simple upper bound on the worst-case expected loss, and then to minimize this upper bound over some tractable class of possible estimators; the resulting optimized estimator will then be guaranteed to possess good worst-case properties. Clearly we want this upper bound to be as tight as possible, and the space of allowed estimators to be as large as possible, while still allowing easy minimization. The approach taken here is to develop bounds which are convex in the estimator, and to allow the estimators to range over a large convex space; this implies that the minimization problem is tractable by descent methods, since no non-global local minima exist.

We begin by defining the class of estimators we will be minimizing over: $\hat{p}$ of the form

$$\hat{p}_i = \frac{g(n_i)}{\sum_{i=1}^{m} g(n_i)},$$

with $n_i$ defined as the number of samples observed in bin $i$ and the constants $g_j \equiv g(j)$ taking values in the $(N+1)-$dimensional convex space $g_j \geq 0$; note that normalization of the estimated distribution is automatically enforced. The "add-constant" estimators, $g_j = \frac{j+\alpha}{N+m\alpha}, \alpha > 0$, are an important special case (7).

After some rearrangement, the expected KL loss for these estimators satisfies

$$
\begin{aligned}
E_{\vec{p}}(L(\vec{p}, \hat{p})) &= E_{\vec{p}}\left(\sum_{i=1}^{m} p_i \log \frac{p_i}{\hat{p}_i}\right) \\
&= \sum_i \left(-H(p_i) + \sum_{j=0}^{N}(-\log g_j)p_i B_{N,j}(p_i)\right) + E_{\vec{p}}\left(\log \sum_{k=1}^{m} g(n_k)\right) \\
&\leq \sum_i \left(-H(p_i) + \sum_j(-\log g_j)p_i B_{N,j}(p_i)\right) + E_{\vec{p}}\left(-1 + \sum_k g(n_k)\right) \\
&= \sum_i f(p_i);
\end{aligned}
$$

we have abbreviated $\vec{p}$ the true underlying distribution, the entropy function

$$H(t) = -t \log t,$$

the binomial functions

$$B_{N,j}(t) = \binom{N}{j} t^j (1-t)^{N-j},$$

and

$$f(t) = -H(t) - t + \sum_j (g_j - t \log g_j) B_{N,j}(t).$$

Equality holds iff $\sum_k g(n_k)$ is constant almost surely (as is the case, e.g., for any add-constant estimator).

We have two distinct simple bounds on the above: first, the obvious

$$\sum_{i=1}^{m} f(p_i) \leq m \max_{0 \leq t \leq 1} f(t),$$

which generalizes the bound considered in (*7*) (where a similar bound was derived asymptotically as $N \to \infty$ for $m$ fixed, and applied only to the add-constant estimators), or

$$\sum_i f(p_i) \leq \left( m \max_{0 \leq t \leq 1/m} f(t) \right) + \left( \max_{1/m \leq t \leq 1} \frac{f(t)}{t} \right),$$

which follows easily from $\sum_i p_i = 1$; see (*11*) for a proof. The above maxima are always achieved, by the compactness of the intervals and the continuity of the binomial and entropy functions. Again, the key point is that these bounds are uniform over all possible underlying $p$ (that is, they bound the worst-case error).

Why two bounds? The first is nearly tight for $N >> m$ (it is actually asymptotically possible to replace $m$ with $m - 1$ in this limit, due to the fact that $p_i$ must sum to one; see (*7, 8*)), but grows linearly with $m$ and thus cannot be tight for $m$ comparable to or larger than $N$. In particular, the optimizer doesn't depend on $m$, only $N$ (and hence the bound can't help but behave linearly in $m$). The second bound is much more useful (and, as we show below, tight) in the data-sparse regime $N << m$.

The resulting minimization problems have a polynomial approximation flavor: we are trying to find an optimal set of weights $g_j$ such that the sum in the definition of $f(t)$ (a polynomial in $t$) will be as close to $H(t) + t$ as possible. In this sense our approach is nearly identical to that recently followed for bounding the bias in the entropy estimation case (*11, 12*). There are three key differences, however: the term penalizing the variance in the entropy case is missing here, the approximation only has to be good from above, not from below as well (both making the problem easier), and the approximation is nonlinear, instead of linear, in $g_j$ (making the problem harder). Indeed, we will see below that the entropy estimation problem is qualitatively easier than the estimation of the full distribution, despite the entropic form of the KL loss.

**Smooth minimization algorithm**

In the next subsections, we develop methods for minimizing these bounds as a function of $g_j$ (that is, for choosing estimators with good worst-case properties). The first key point is that the bounds involve maxima over a collection of convex functions in $g_j$, and hence the bounds are convex in $g_j$; since the coefficients $g_j$ take values in a convex set, no non-global local minima exist, and the global mimimum can be found by simple descent procedures.

One complicating factor is that the bounds are nondifferentiable in $g_j$: while methods for direct minimization of this type of $L_\infty$ error exist (*13*), they require that we track the location in $t$ of the maximal error; since this argmax can jump discontinuously as a function of $g_j$, this interior maximization loop can be time-consuming. A more efficient solution is given by approximating this nondifferentiable objective function by smooth functions which retain the convexity of the original objective. We employ a Laplace approximation (albeit in a different direction than usual): use the fact that

$$\max_{t \in A} h(t) = \lim_{q \to \infty} \frac{1}{q} \log \int_{t \in A} e^{qh(t)}$$

for continuous $h(t)$ and compact $A$; thus, letting $h(t) = f(t)$, we can minimize

$$U_q(\{g_j\}) \equiv \int_0^1 e^{qf(t)} dt,$$

or

$$V_q(\{g_j\}) \equiv \log\left(\int_0^{1/m} e^{qmf(t)}dt\right) + \log\left(\int_{1/m}^1 e^{q\frac{f(t)}{t}}dt\right),$$

for $q$ increasing; these new objective functions are smooth, with easily-computable gradients, and are still convex, since $f(t)$ is convex in $g_j$, convex functions are preserved under convex, increasing maps (i.e., the exponential), and sums of convex functions are convex. (In fact, since $U_q$ is strictly convex in $g$ for any $q$, the minima are unique, which to our knowledge is not necessarily the case for the original minimax problem.) It is easy to show that any limit point of the sequence of minimizers of the above problems will minimize the original problem; applying conjugate gradient descent for each $q$, with the previous minimizer as the seed for the minimization in the next largest $q$, worked well in practice.

### Initialization; connection to Laplace estimator

It is now useful to look for suitable starting points for the minimization. For example, for the first bound, approximate the maximum by an integral, that is, find $g_j$ to minimize

$$m\int_0^1 dt\left(-H(t) - t + \sum_j (g_j - t\log g_j)B_{N,j}(t)\right).$$

(Note that this can be thought of as the limit of the above $U_q$ minimization problem as $q \to 0$, as can be seen by expanding the exponential.) The $g_j$ that minimizes this approximation to the upper bound is trivially derived as

$$g_j = \frac{\int_0^1 tB_{N,j}(t)dt}{\int_0^1 B_{N,j}(t)dt} = \frac{\beta(j+2, N-j+1)}{\beta(j+1, N-j+1)} = \frac{j+1}{N+2},$$

with $\beta(a,b) = \int_0^1 t^{a-1}(1-t)^{b-1}dt$ defined as usual. The resulting estimator $\hat{p}$ agrees exactly with "Laplace's estimator," the add-$\alpha$ estimator with $\alpha = 1$. Note, though, that to derive this $g_j$, we completely ignore the first two terms $(-H(t) - t)$ in the upper bound, and the resulting estimator can therefore be expected to be suboptimal (in particular, the $g_j$ will be chosen too large, since $-H(t) - t$ is strictly decreasing for $t < 1$). Indeed, we find that add-$\alpha$ estimators with $\alpha < 1$ provide a much better starting point for the optimization, as expected given (7, 8). (Of course, for $N/m$ large enough an asymptotically optimal estimator is given by the perturbed add-constant estimator of (8), and none of this numerical optimization is necessary.) In the limit as $c = N/m \to 0$, we will see below that a better initialization point is the add-$\alpha$ estimator with parameter $\alpha \approx H(c) = -c\log c$.

### Fixed-point algorithm

On examining the gradient of the above problems with respect to $g_j$, a fixed-point algorithm may be derived. We have, for example, that

$$\frac{\partial U}{\partial g_j} = \int_0^1 dt\left(1 - \frac{t}{g_j}\right)e^{qf(t)}B_{N,j}(t);$$

thus, analogously to the $q \to 0$ case above, a simple update is given by

$$g_j^1 = \frac{\int_0^1 te^{qf^0(t)}B_{N,j}(t)dt}{\int_0^1 e^{qf^0(t)}B_{N,j}(t)dt},$$

which effectively corresponds to taking the mean of the binomial function $B_{N,j}$, weighted by the "importance" term $e^{qf(t)}$, which in turn is controlled by the proximity of $t$ to the maximum of $f^0(t)$ for $q$ large. While this is an attractive strategy, conjugate gradient descent proved to be a more stable algorithm in our hands.

## Lower bounds

Once we have found an estimator with good worst-case error, we want to compare its performance to some well-defined optimum. To do this, we obtain lower bounds on the worst-case performance of *any* estimator (not just the class of $\hat{p}$ we minimized over in the last section). Once again, we will derive a family of bounds indexed by some parameter $\alpha$, and then optimize over $\alpha$.

Our lower bounds are based on the well-known fact that, for any proper prior distribution, the average (Bayesian) loss is less than or equal to the maximum (worst-case) loss. The most convenient class of priors to use here are the Dirichlet priors. Thus we will compute the average KL error under any Dirichlet distribution (interesting in its own right), then maximize over the possible Dirichlet priors (that is, find the "least favorable" Dirichlet prior) to obtain the tightest lower bound on the worst-case error; importantly, the resulting bounds will be nonasymptotic (that is, valid for all $N$ and $m$). This approach therefore generalizes the asymptotic lower bound used in (*7*), who examined the KL loss under the special case of the uniform Dirichlet prior. See also (*4*) for direct application of this idea to bound the average code length, and (*14*), who derived a lower bound on the average KL loss, again in the uniform Dirichlet case.

We compute the Bayes error as follows. First, it is well-known (e.g., (*9, 14*)) that the KL-Bayes estimate of $\vec{p}$ given count data $\vec{n}$ (under any prior, not just the Dirichlet) is the posterior mean (interestingly, the KL loss shares this property with the squared error); for the Dirichlet prior with parameter $\vec{\alpha}$, this conditional mean has the particularly simple form

$$E_{Dir(\vec{\alpha}|\vec{n})}\vec{p} = \frac{\vec{\alpha} + \vec{n}}{\sum_i \alpha_i + n_i},$$

with $Dir(\vec{\alpha}|\vec{n})$ denoting the $Dir(\vec{\alpha})$ density conditioned on data $\vec{n}$. Second, it is straightforward to show (*14*) that the conditional average KL error, given this estimate, has an appealing form: the entropy at the conditional mean minus the conditional mean entropy (one can easily check the strict positivity of this average error via the concavity of the vector entropy function $H(\vec{p}) = -\sum_i p_i \log p_i$). Thus we can write the average loss as

$$E_{Dir(\vec{\alpha})}\left[H(\frac{\vec{\alpha}+\vec{n}}{\sum_i \alpha_i+n_i}) - E_{Dir(\vec{\alpha}|\vec{n})}H(\vec{p})\right] = \sum_i E_{Dir(\vec{\alpha})}\left[H(\frac{\alpha_i+n_i}{N+\sum_i \alpha_i}) - E_{Dir(\vec{\alpha}+\vec{n})}H(p_i)\right],$$

where the inner averages over $\vec{p}$ are under the Dirichlet distribution and the outer averages over $\vec{n}$ and $n_i$ are under the corresponding Dirichlet-multinomial or Dirichlet-binomial mixtures (i.e., multinomials whose parameter $\vec{p}$ is itself Dirichlet distributed); we have used linearity of the expectation, $\sum_i n_i = N$, and $Dir(\vec{\alpha}|\vec{n}) = Dir(\vec{\alpha} + \vec{n})$. Evaluating the right-hand side of the above, in turn, requires the formula

$$-E_{Dir(\alpha)}H(p_i) = \frac{\alpha_i}{\sum_i \alpha_i}\left(\psi(\alpha_i + 1) - \psi(1 + \sum_i \alpha_i)\right),$$

with $\psi(t) = \frac{d}{dt}\log\Gamma(t)$; recall that $\psi(t+1) = \psi(t) + \frac{1}{t}$. All of the above may thus be easily computed numerically for any $N, m$, and $\vec{\alpha}$; to simplify, however, we will restrict $\vec{\alpha}$ to be constant, $\vec{\alpha} = (\alpha, \alpha, \dots, \alpha)$. This symmetrizes the above formulae; we can replace the outer sum with multiplication by $m$, and substitute $\sum_i \alpha_i = m\alpha$. Finally, abbreviating $K = N + m\alpha$, we have that the worst-case error is bounded below by:

$$\frac{m}{K}\sum_{j=0}^{N} p_{\alpha,m,N}(j)(j+\alpha)\left(-\log\frac{j+\alpha}{K} + \psi(j+\alpha) + \frac{1}{j+\alpha} - \psi(K) - \frac{1}{K}\right), \quad (1)$$

with $p_{\alpha,m,N}(j)$ the beta-binomial distribution

$$p_{\alpha,m,N}(j) = \binom{N}{j}\frac{\Gamma(m\alpha)\Gamma(j+\alpha)\Gamma(K-(j+\alpha))}{\Gamma(K)\Gamma(\alpha)\Gamma(m\alpha-\alpha)}.$$

This lower bound is valid for all $N, m$, and $\alpha$, and can be optimized numerically in the (scalar) parameter $\alpha$ in a straightforward manner.

## Asymptotic analysis

In this section, we aim to understand some of the implications of the rather complicated expressions above, by analyzing them in some simplifying limits. Due to space constraints, we can only sketch the proof of each of the following statements.

**Proposition 1.** *Any add-$\alpha$ estimator, $\alpha > 0$, is uniformly KL-consistent if $N/m \to \infty$.*

This is a simple generalization of a result of (*7*), who proved consistency for the special case of $m$ fixed and $N \to \infty$; the main point here is that $N/m$ is allowed to tend to infinity arbitrarily slowly. The result follows on utilizing our first upper bound (the main difference between our analysis and that of (*7*) is that our bound holds for all $m, N$, whereas (*7*) focuses on the asymptotic case) and noting that $\max_{0 \le t \le 1} f(t) = O(1/N)$ for $f(t)$ defined by any add-constant estimator; hence our upper bound is uniformly $O(m/N)$. To obtain the $O(1/N)$ bound, we plug in the add-constant $g_j = (j + \alpha)/N$:

$$f(t) = \alpha/N + t \left( \log t - \sum_j (\log \frac{j+\alpha}{N}) B_{N,j}(t) \right).$$

For $t$ fixed, an application of the delta method implies that the sum looks like $\log(t + \frac{\alpha}{N}) - \frac{1-t}{2Nt}$; an expansion of the logarithm, in turn, implies that the right-hand side converges to $\frac{1}{2N}(1 - t)$, for any fixed $\alpha > 0$. On a $1/N$ scale, on the other hand, we have

$$Nf(\frac{t}{N}) = \alpha + t \left( \log t - \sum_j \log(j + \alpha) B_{N,j}(\frac{t}{N}) \right),$$

which can be uniformly bounded above. In fact, as demonstrated by (*7*), the binomial sum on the right-hand side converges to the corresponding Poisson sum; interestingly, a similar Poisson sum plays a key role in the analysis of the entropy estimation case in (*12*). $\square$

A converse follows easily from the lower bounds developed above:

**Proposition 2.** *No estimator is uniformly KL-consistent if $\limsup N/m < \infty$.*

Of course, it is intuitively clear that we need many more than $m$ samples to estimate a distribution on $m$ bins; our contribution here is a quantitative asymptotic lower bound on the error in the data-sparse regime. (A simpler but slightly weaker asymptotic bound may be developed from the lower bound given in (*14*).) Once again, we contrast with the entropy estimation case, where consistent estimators do exist in this regime (*12*).

We let $N, m \to \infty, N/m \to c, 0 < c < \infty$. The beta-binomial distribution has mean $N/m$ and converges to a non-degenerate limit, which we'll denote $p_{\alpha,c}$, in this regime. Using Fatou's lemma and $\psi(t) = \log(t) - \frac{1}{2t} + O\left(t^{-2}\right)$, $t \to \infty$, we obtain the asymptotic lower bound

$$\frac{1}{c+\alpha} \sum_{j=0}^{\infty} p_{\alpha,c}(j)(\alpha + j) \left( -\log(\alpha + j) + \psi(\alpha + j) + \frac{1}{\alpha + j} \right) > 0. \quad \square$$

Also interestingly, it is easy to see that our lower bound behaves as $\frac{m-1}{2N}(1 + o(1))$ as $N/m \to \infty$ for any fixed positive $\alpha$ (since in this case $\sum_{j=0}^{k} p_{\alpha,m,N}(j) \to 0$ for any fixed finite $k$). Thus, comparing to the upper bound on the minimax error in (*8*), we have the somewhat surprising fact that:

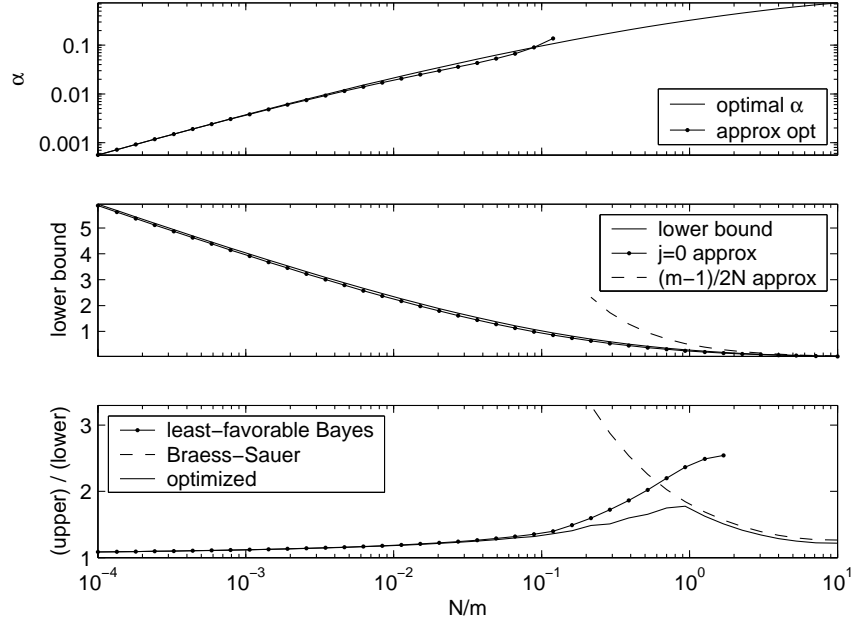

Figure 1: Illustration of bounds and asymptotic results. $N = 100$, $m$ varying. **a**. Numerically- and theoretically-obtained optimal (least-favorable) $\alpha$, as a function of $c = N/m$; note close agreement. **b**. Numerical lower bounds and theoretical approximations; note the log-linear growth as $c \to 0$. The $j = 0$ approximation is obtained by retaining only the $j = 0$ term of the sum in the lower bound (1); this approximation turns out to be sufficiently accurate in the $c \to 0$ limit, while the $(m-1)/2N$ approximation is tight as $c \to \infty$. **c**. Ratio comparison of upper to lower bounds. Dashed curve is the ratio obtained by plugging the asymptotically optimal estimator due to Braess-Sauer (8) into our upper bound; solid-dotted curve numerically least-favorable Dirichlet estimator; black solid curve optimized estimator. Note that curves for optimized and Braess-Sauer estimators are in constant proportion, since bounds are independent of $m$ for $c$ large enough. Most importantly, note that optimized bounds are everywhere tight within a factor of 2, and asymptotically tight as $c \to \infty$ or $c \to 0$.

**Proposition 3.** *Any fixed-$\alpha$ Dirichlet prior is asymptotically least-favorable as $\frac{N}{m} \to \infty$.*

This generalizes Theorem 2 of (7) (and in fact, an alternate proof can be constructed on close examination of Krichevskiy's proof of that result).

Finally, we examine the optimizers of the bounds in the data-sparse limit, $c = N/m \to 0$.

**Proposition 4.** *The least-favorable Dirichlet parameter is given by $H(c)$ as $c \to 0$; the corresponding Bayes estimator also asymptotically minimizes the upper bound (and hence the bounds are asymptotically tight in this limit). The maximal and average errors grow as $-log(c)(1 + o(1))$, $c \to 0$.*

This is our most important asymptotic result. It suggests a simple and interesting rule of thumb for estimating distributions in this data-sparse limit: use the add-$\alpha$ estimator with $\alpha = H(c)$. When the data are very sparse ($c$ sufficiently small) this estimator is optimal; see Fig. 1 for an illustration. The proof, which is longer than those of the above results but still fairly straightforward, has been omitted due to space constraints.

## Discussion

We have omitted a detailed discussion of the form of the estimators which numerically minimize the upper bounds developed here; these estimators were empirically found to be perturbed add-constant estimators, with $g_j$ growing linearly for large $j$ but perturbed downward in the approximate range $j < 10$. Interestingly, in the heavily-sampled limit $N >> m$, the minimizing estimator provided by ($8$) again turns out to be a perturbed add-constant estimator. Further details will be provided elsewhere.

We note an interesting connection to the results of ($9$), who find that $1/m$ scaling of the add-constant parameter $\alpha$ is empirically optimal for for an entropy estimation application with large $m$. This $1/m$ scaling bears some resemblance to the optimal $H(c)$ scaling that we find here, at least on a logarithmic scale (Fig. 1a); however, it is easy to see that the extra $-\log(c)$ term included here is useful. As argued in ($3$), it is a good idea, in the data-sparse limit $N << m$, to assign substantial probability mass to bins which have not seen any data samples. Since the total probability assigned to these bins by any add-$\alpha$ estimator scales in this limit as $P(unseen) = m\alpha/(N + m\alpha)$, it is clear that the choice $\alpha \sim 1/m$ decays too quickly.

Finally, we note an important direction for future research: the upper bounds developed here turn out to be least tight in the range $N \approx m$, when the optimum in the bound occurs near $t = 1/m$; in this case, our bounds can be loose by roughly a factor of two (exactly the degree of looseness we found in Fig. 1c). Thus it would be quite worthwhile to explore upper bounds which are tight in this $N \approx m$ range.

**Acknowledgements**: We thank Z. Ghahramani and D. Mackay for helpful conversations; LP is supported by an International Research Fellowship from the Royal Society.

## References

1. D. Mackay, L. Peto, *Natural Language Engineering* **1**, 289 (1995).
2. N. Friedman, Y. Singer, *NIPS* (1998).
3. A. Orlitsky, N. Santhanam, J. Zhang, *Science* **302**, 427 (2003).
4. T. Cover, *IEEE Transactions on Information Theory* **18**, 216 (1972).
5. R. Krichevsky, V. Trofimov, *IEEE Transactions on Information Theory* **27**, 199 (1981).
6. D. Braess, H. Dette, *Sankhya* **66**, 707 (2004).
7. R. Krichevsky, *IEEE Transactions on Information Theory* **44**, 296 (1998).
8. D. Braess, T. Sauer, *Journal of Approximation Theory* **128**, 187 (2004).
9. T. Schurmann, P. Grassberger, *Chaos* **6**, 414 (1996).
10. I. Nemenman, F. Shafee, W. Bialek, *NIPS* **14** (2002).
11. L. Paninski, *Neural Computation* **15**, 1191 (2003).
12. L. Paninski, *IEEE Transactions on Information Theory* **50**, 2200 (2004).
13. G. Watson, *Approximation theory and numerical methods* (Wiley, Boston, 1980).
14. D. Braess, J. Forster, T. Sauer, H. Simon, *Algorithmic Learning Theory* **13**, 380 (2002).
